# Dynamic Visual Attention: Searching for coding length increments

**Xiaodi Hou**[1,2] **and Liqing Zhang**[1] *

[1]Department of Computer Science and Engineering, Shanghai Jiao Tong University
No. 800 Dongchuan Road, 200240, China
[2]Department of Computation and Neural Systems, California Institute of Technology
MC 136-93, Pasadena, CA, 91125, USA
xhou@caltech.edu, zhang-lq@sjtu.edu.cn

## Abstract

A visual attention system should respond placidly when common stimuli are presented, while at the same time keep alert to anomalous visual inputs. In this paper, a dynamic visual attention model based on the rarity of features is proposed. We introduce the Incremental Coding Length (ICL) to measure the perspective entropy gain of each feature. The objective of our model is to maximize the entropy of the sampled visual features. In order to optimize energy consumption, the limit amount of energy of the system is re-distributed amongst features according to their Incremental Coding Length. By selecting features with large coding length increments, the computational system can achieve attention selectivity in both static and dynamic scenes. We demonstrate that the proposed model achieves superior accuracy in comparison to mainstream approaches in static saliency map generation. Moreover, we also show that our model captures several less-reported dynamic visual search behaviors, such as attentional swing and inhibition of return.

## 1   Introduction

Visual attention plays an important role in the human visual system. This voluntary mechanism allows us to allocate our sensory and computational resources to the most valuable information embedded in the vast amount of incoming visual data. In the past decade, we have witnessed the success of a number of computational models on visual attention (see [6] for a review). Many of these models analyze static images, and output "saliency maps", which indicate the probability of eye fixations. Models such as [3] and [4] have tremendously boosted the correlation between eye fixation data and saliency maps.

However, during the actual continuous perception process, important dynamic behaviors such as the sequential order of attended targets, shifts of attention by saccades, and the inhibitory mechanism that precludes us from looking at previously observed targets, are not thoroughly discussed in the research on visual attention. Rather than contributing to the accuracy of saliency map generation, we instead consider alternative approaches to understand visual attention: is there a model that characterizes the ebbs and flows of visual attention?

Up to the present, this question is not comprehensively answered by existing models. Algorithms simulating saccades in some attention systems [23, 7] are designed for engineering expediency rather than scientific investigation. These algorithms are not intended to cover the full spectrum of dynamic properties of attention, nor to provide a convincing explanation of the continuous nature of attention behaviors.

In this paper, we present a novel attention model that is intrinsically continuous. Unlike space-based models who take discrete frames of images as the elementary units, our framework is based on continuous sampling of features. Inspired by the principle of predictive coding [9], we use the concept of energy to explain saliency, feature response intensity, and the appropriation of computational resources in one unified framework. The appropriation of energy is based on the Incremental Coding Length, which indicates the rarity of a feature. As a result, stimuli that correlate to rarely activated features will receive the highest energy, and become salient. Since the proposed model is temporally continuous, we can demonstrate a series of simulations of dynamic attention, and provide plausible explanations of previously unexamined behaviors.

## 1.1 Space and Feature Based Attention

Many of the bottom-up visual attention models follow the Koch and Ullman framework [10]. By analyzing feature maps that topographically encode the spatial homogeneity of features, an algorithm can detect the local irregularities of the visual input. This paradigm explains the generation of attention from a one-shot observation of an image. However, several critical issues may be raised when this framework is applied to continuous observations (e.g. video). First, space-based attention itself cannot interpret ego-motion. Additional coordinate transformation models are required to translate spatial cues between two different frames. Second, there are attention mechanisms that operate after the generation of saliency, such as attentional modulation [19], and Inhibition of Return (IOR) [8]. The initial space-based framework is not likely to provide a convincing explanation to these mechanisms.

In addition to saliency based on local irregularity, recent investigations in V4 and MT cortical areas demonstrate that attention can also be elicited by particular features [13, 18]. In the field of computational models, explorations that are biased by features are also used in task-dependent spatial saliency analysis [16]. The emerging evidence in feature-driven attention has encouraged us to propose a pure feature-based attention model in parallel with the space-based feature map paradigm.

## 1.2 On the Cause of Attention

Finding "irregular patterns" as a criterion for attention is widely used in computational models. In a more rigid form, saliency can be defined by the residuals of Difference of Gaussian filter banks [7], regions with maximal self-information [3], or most discriminant center-surround composition [4]. However, all of these principles do little to address the cause of saliency mechanisms in the brain.

At the level of computation, we cannot attribute the formation of attention to functional advantages such as foraging for foods [6]. In this paper, we hypothesize that visual attention is driven by the predictive coding principle, that is, the optimization of metabolic energy consumption in the brain. In our framework, the behavior of attention is explained as a consequence of an actively-searching observer who seeks a more economical neural code to represent the surrounding visual environment.

## 2 The Theory

Motivated by the sparse coding strategy [15] discovered in primary visual cortex, we represent an image patch as a linear combination of sparse coding basis functions, which are referred as features. The activity ratio of a feature is its average response to image patches over time and space. The activity of the feature ensemble is considered as a probability function. We evaluate each feature with respect to its *Incremental Coding Length* (ICL). The ICL of $i^{th}$ feature is defined as the ensemble's entropy gain during the activity increment of $i^{th}$ feature. In accordance with the general principle of predictive coding [17], we redistribute energy to features according to their ICL contribution: frequently activated features receive less energy than rarer features. Finally, the saliency of a region is obtained by summing up the activity of all features at that region.

## 2.1 Sparse Feature Representation

Experimental studies [15] have shown that the receptive fields of simple-cells in the primary visual cortex produce a sparse representation. With standard methods [2], we learn a set of basis functions that yields a sparse representation of natural image patches. These basis functions are used as

features in the analysis of attention. Specifically, we use 120000 $8 \times 8$ RGB image patches from natural scenes for training. A set of $8 \times 8 \times 3 = 192$ basis functions is obtained. (See Fig. 1).

Let $\mathbf{A}$ be the sparse basis, where $\mathbf{a}^i$ is the $i^{th}$ basis function. Let $\mathbf{W} = \mathbf{A}^{-1}$ be the bank of filter functions, where $\mathbf{W} = [\mathbf{w}_1, \mathbf{w}_2, \ldots, \mathbf{w}_{192}]^\top$. Each row vector $\mathbf{w}_j$ of $\mathbf{W}$ can be considered as a linear filter to the image patch.

The sparse representation $\mathbf{s}$ of an image patch is its response to all filter functions. Given a vectorized image $\mathbf{x}$, we have $\mathbf{s} = \mathbf{W}\mathbf{x}$. Since each basis function represents a structural primitive, in the cortex representation of natural images, only a small population of neurons are activated at one time. Considering the energy consumed by neural activity in the brain, this sparse coding strategy is advantageous [11].

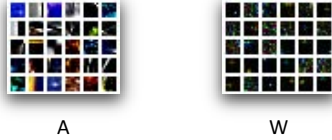

A           W

Figure 1: First 30 components of the basis functions $\mathbf{A}$ and the corresponding filter functions $\mathbf{W}$ are shown in this figure.

## 2.2   The Incremental Coding Length

In contrast to the long-term evolution of sparse representation, which reflects the general statistics of nature, short-term habituations, such as potentiation of synaptic strengths, occur during brief observations in a particular environment. In order to evaluate the immediate energy changes in the cortex, some previous work has analyzed the information representation and coding in early visual system [20, 21, 1]. Guided by the insights behind predictive coding [17], we propose the Incremental Coding Length (ICL) as a computational principle based on features. This principle aims to optimize the immediate energy distribution in the system in order to achieve an energy-economic representation of its environment.

The activity ratio $p_i$ for $i^{th}$ feature is defined as its relative response level over a sequence of sampling. Given the sample matrix $\mathbf{X} = [\mathbf{x}^1, \mathbf{x}^2, \ldots, \mathbf{x}^k, \ldots]$, where $\mathbf{x}^k$ is an vectorized image patch, we can compute the activity ratio $p_i$ as:

$$p_i = \frac{\sum_k \mid \mathbf{w}_i \mathbf{x}^k \mid}{\sum_i \sum_k \mid \mathbf{w}_i \mathbf{x}^k \mid}. \tag{1}$$

Furthermore, we denote $\mathbf{p} = [p_1, p_2, \ldots]^\top$ as the probability function of feature activities. Note that the activity ratio and the energy are abstract values that reflect the statistics of features. Wiring this structure at the neuronal level goes beyond the scope of this paper. However, studies [13] have suggested evidence of a population of neurons that is capable of generating a representation for intermodal features. In our implementation, the distribution $\mathbf{p}$ addresses the computational properties of this putative center.

Since the visual information is jointly encoded by all features, the most efficient coding strategy should *make equal use of all possible feature response levels*. To achieve this optimality, the model needs to maximize the entropy $H(\mathbf{p})$. Since $\mathbf{p}$ is determined by the samples $\mathbf{X}$, it is possible for a system to actively bias the sampling process in favor of maximizing information transmission.

At a certain point of time, the activity ratio distribution is $\mathbf{p}$. We consider a new excitation to feature $i$, which will add a variation $\varepsilon$ to $p_i$, and change the whole distribution. The new distribution $\hat{\mathbf{p}}$ is:

$$\hat{p}_j = \begin{cases} \frac{p_j + \varepsilon}{1 + \varepsilon}, & j = i \\ \frac{p_j}{1 + \varepsilon}, & j \neq i \end{cases}$$

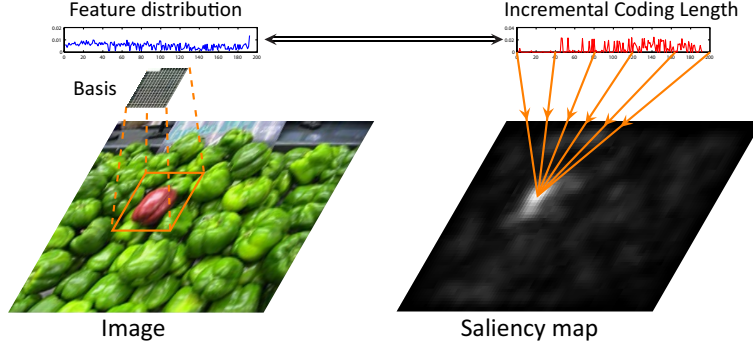

Feature distribution

Incremental Coding Length

Basis

Image

Saliency map

Figure 2: The framework of feature-based selective attention.

This variation therefore changes the entropy of feature activities. The change of entropy with respect to the feature activity probability increment is:

$$\frac{\partial H(\mathbf{p})}{\partial p_i} = -\frac{\partial p_i \log p_i}{\partial p_i} - \frac{\partial \sum_{j \neq i} p_j \log p_j}{\partial p_i} = -1 - log p_i - \frac{\partial \sum_{j \neq i} p_j \log p_j}{\partial p_i},$$

where:

$$\frac{\partial \sum_{j \neq i} p_j \log p_j}{\partial p_i} = H(\mathbf{p}) - 1 + p_i + p_i \log p_i,$$

Accordingly, we define the Incremental Coding Length (ICL) to be:

$$\text{ICL}(p_i) = \frac{\partial H(\mathbf{p})}{\partial p_i} = -H(\mathbf{p}) - p_i - \log p_i - p_i \log p_i \quad (2)$$

### 2.3 Energy Redistribution

We define the *salient feature set* $\mathcal{S}$ as: $\mathcal{S} = \{i \mid \text{ICL}(p_i) > 0\}$. The partition $\{\mathcal{S}, \bar{\mathcal{S}}\}$ tells us whether successive observations of feature $i$ would increase $H(\mathbf{p})$. In the context of visual attention, the intuition behind the salient feature set is straightforward: A feature is salient *only when succeeding activations of that feature can offer entropy gain to the system.*

Within this general framework of feature-level optimization, we can redistribute the energy among features. The amount of energy received by each feature is denoted $d_i$. Non-salient features are automatically neglected by setting $d_k = 0 \quad (k \in \bar{\mathcal{S}})$. For features in the salient feature set, let:

$$d_i = \frac{\text{ICL}(p_i)}{\sum_{j \in \mathcal{S}} \text{ICL}(p_j)}, \quad (\text{if } i \in \mathcal{S}). \quad (3)$$

Finally, given an image $\mathbf{X} = [\mathbf{x}^1, \mathbf{x}^2, \ldots, \mathbf{x}^n]$, we can quantify the saliency map $\mathbf{M} = [\mathbf{m}_1, \mathbf{m}_2, \ldots, \mathbf{m}_n]$ as:

$$\mathbf{m}_k = \sum_{i \in \mathcal{S}} d_i \mathbf{w}_i \mathbf{x}^k. \quad (4)$$

In Eq. 4, we notice that the saliency of a patch is not constant. It is determined by the distribution of $\boldsymbol{p}$, which can be obtained by sampling the environment over space and time.

According to Eq. 4, we notice that the saliency of a patch may vary over time and space. An intuitive explanation to this property is the contextual influence: under different circumstances, "salient features" are defined in different manners to represent the statistical characteristics of the immediate environment.

# 3 The Experiment

We proposed a framework that explains dynamic visual attention as a process that spends limited available energy preferentially on rarely-seen features. In this section, we examine experimentally the behavior of our attention model.

## 3.1 Static Saliency Map Generation

By sequentially sampling over all possible image patches, we calculate the feature distribution of a static image and generate the corresponding saliency map. These maps are then compared with records of eye fixations of human subjects. The accuracy of an algorithm is judged by the area under its ROC curve.

We use the fixation data collected by Bruce et al. [3] as the benchmark for comparison. This data set contains the eye fixation records from 20 subjects for the full set of 120 images. The images are down-sampled to an appropriate scale ($86 \times 64$, $\frac{1}{4}$ of the original size). The results for several models are indicated below. Due to a difference in the sampling density used in drawing the ROC curve, the listed performance is slightly different (about 0.003) from that given in [3] and [4]. The algorithms, however, are all evaluated using the same benchmark and their relative performance should be unaffected. Even though it is not designed for static saliency map generation, our model achieves the best performance among mainstream approaches.

Table 1: Performances on static image saliency

| Itti et al. [7] | Bruce et al. [3] | Gao et al. [4] | Our model |
|---|---|---|---|
| 0.7271 | 0.7697 | 0.7729 | **0.7928** |

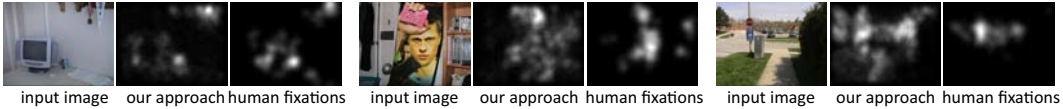

input image   our approach   human fixations   input image   our approach   human fixations   input image   our approach   human fixations

Figure 3: Some examples of our experimental images.

## 3.2 Dynamic Saliency on Videos

A distinctive property of our model is that it is updated online. As proposed in Eq. 2, ICL is defined by the feature activity ratio distribution. This distribution can be defined over space (when sampling within one 2-D image) as well as over time (when sampling over a sequence of images). The temporal correlation among frames can be considered as a Laplacian distribution. Accordingly, at the $t^{th}$ frame, the cumulative activity ratio distribution $\mathbf{p}^t$ yields:

$$\mathbf{p}^t = \frac{1}{Z} \sum_{\tau=0}^{t-1} \exp(\frac{\tau - t}{\lambda}) \cdot \hat{\mathbf{p}}^\tau, \tag{5}$$

where $\lambda$ is the half life. $\hat{\mathbf{p}}_\tau$ is the feature distribution of the $\tau^{th}$ image. $Z = \int \mathbf{p}^t(x) \mathrm{d}x$ is the normalization factor that ensures $\mathbf{p}^t$ is a probability distribution.

In video saliency analysis, one of the potential challenges comes from simultaneous movements of the targets and self-movements of the observer. Since our model is feature-based, spatial movements of an object or changing perspectives will not dramatically affect the generation of saliency maps. In order to evaluate the detection accuracy of our approach under changing environment, we compare the dynamic visual attention model with models proposed in [7] and [5].

In this experiment, we use a similar criterion to that described in [5]. The efficacy of the saliency maps to a videoclip is determined by comparing the response intensities at saccadic locations and random locations. Ideally, an effective saliency algorithm would have high output at locations gazed by observers, and tend not to response in most of the randomly chosen locations.

To quantify this tendency of selectivity, we first compute the distribution of saliency value at human saccadic locations $q_s$ and the distribution at random locations $q_r$. Then, KL divergency is used to measure their dissimilarity. Higher the KL divergency is, more easily a model can discriminate human saccadic locations in the image.

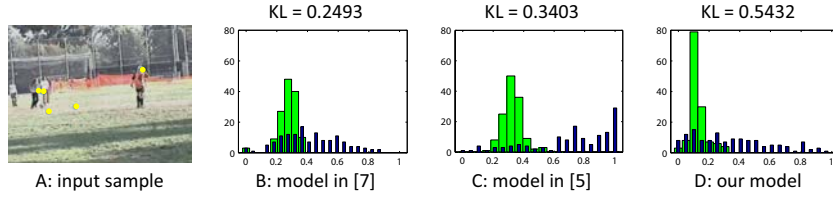

Figure 4: The eye-track records and the video is obtained from [5]. This video contains both target movements and self-movements. In this video, 137 saccades (yellow dots in figure A) are collected. Given the sequence of generated saliency maps, we can obtain the saliency distribution at human saccade locations (narrow blue bars), and random locations (wide green bars). The KL-divergency of these two distribution indicates the performance of each model.

### 3.3 Dynamic Visual Search

We are particularly interested in the dynamic behaviors of attention. Reported by researchers in neurobiological experiments, an inhibitory effect was aroused after sustained attention [12]. This mechanism is referred as Inhibition of Return (IOR) [8]. Research on the cumulative effects of attention [24] has suggested that the dynamics of visual search have broad implications for scene perception, perceptual learning, automaticity, and short term memory. In addition, as a mechanism that prevents an autonomous system from being permanently attracted to certain salient spots and thereby to facilitate productive exploration, the computational modeling of IOR is of practical value in AI and robotics. Previous computational models such as [22, 7] implemented the IOR in a spatially-organized, top-down manner, whereas our model samples the environment online and is driven by data in a bottom-up manner. Spontaneous shifts of attention to new visual cues, as well as the "refusal of perception" behavior arise naturally as consequences of our active search model. Moreover, unlike the spatial "inhibitory masking" approach in [7], our model is feature-based and is therefore free from problems caused by spatial coordinate transformations.

#### 3.3.1 Modeling Sensory Input

The sensory structure of the human retina is not uniform. The resolution of perception decreases when eccentricity increases. In order to overcome the physical limitations of the retina, an overt eye movement is made so that the desired visual stimuli can be mapped onto the foveal region. Similar to the computational approximations in [14], we consider the fovea sampling bias as a weighted mask $\mathbf{W}$ over the reconstructed saliency map. Let the fovea be located at $(x_0, y_0)$; the saliency at $(x, y)$ is weighted by $\mathbf{W}(x, y)$:

$$\mathbf{W}(x,y) = e^{-\frac{1}{2}\left[(x-x_0)^2+(y-y_0)^2\right]} + \xi. \tag{6}$$

In the experiments, we choose $\xi = 1$.

#### 3.3.2 Overt Eye Movements towards Saliency Targets with Inhibition of Return

In the incremental perception of one static image, our dynamic visual system is guided by two factors. The first factor is the non-homogeneous composition of features in the observed data that fosters feature preferences in the system. The second factor is a foveal structure that allows the system to bias its sampling via overt eye movements. The interplay of these two factors leads to an active visual search behavior that moves towards a maximum entropy equilibrium in the feature distribution. It is also worth noting that these two factors achieve a hysteresis effect that is responsible for Inhibition Of Return (IOR). A recently attended visual region is not likely to regain eye fixation within short interval because of the foveated weighting. This property of IOR is demonstrated by our experiments.

An implementation of our dynamic visual search is shown in the algorithm box.

---

**Dynamic Visual Attention**

1. At time $t$, calculate feature ICL based on $\mathbf{p}^t$
2. Given current eye fixation, generate a saliency map with foveal bias.
3. By a saccade, move eye to the global maximum of the saliency map.
4. Sample top $N$ "informative" (largest ICL) features in fixation neighborhood. (In our experiment, $N = 10$)
5. Calculate $\hat{\mathbf{p}}^t$, update $\mathbf{p}^{t+1}$, and go to Step. 1.

---

It is also worth noting that, when run on the images provided by [3], our dynamic visual attention algorithm demonstrates especially pronounced saccades when multiple salient regions are presented in the same image. Although we have not yet validated these saccades against human retinal data, to our knowledge this sort of "attentional swing" has never been reported in other computational systems.

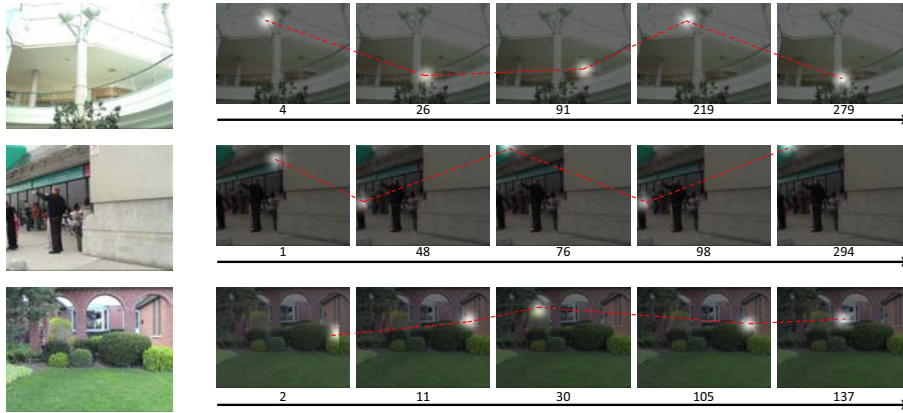

Figure 5: Results on dynamic visual search

## 4 Discussions

A novel dynamic model of visual attention is described in this paper. We have proposed Incremental Coding Length as a general principle by which to distribute energy in the attention system. In this principle, the salient visual cues correspond to unexpected features - according to the definition of ICL, these features may elicit entropy gain in the perception state and are therefore assigned high energy.

To validate this theoretical framework, we have examined experimentally various aspects of visual attention. In experiments comparing with static saliency maps, our model more accurately predicted saccades than did other mainstream models. Because the model updates its state in an online manner, we can consider the statistics of a temporal sequence and our model achieved strong results in video saliency generation. Finally, when feature-based ICL is combined with foveated sampling, our model provides a coherent mechanism for dynamic visual search with inhibition of return.

In expectation of further endeavors, we have presented the following original ideas. 1) In addition to spatial continuity cues, which are demonstrated in other literature, saliency can also be measured using features. 2) By incorporating temporal dynamics, a visual attention system can capture a broad range of novel behaviors that have not successfully been explained by saliency map analysis. And 3) dynamic attention behaviors might quantitatively be explained and simulated by the pursuit of a maximum entropy equilibrium in the state of perception.

# 5   Acknowledgements

We thank Neil Bruce, John Tsotsos, and Laurent Itti for sharing their experimental data. The first author would like to thank Charles Frogner, Yang Cao, Shengping Zhang and Libo Ma for their insightful discussions on the paper. The reviewers' pertinent comments and suggestions also helped to improve the quality of the paper. The work was supported by the National High-Tech Research Program of China (Grant No. 2006AA01Z125) and the National Basic Research Program of China (Grant No. 2005CB724301)

## Footnotes

*http://www.its.caltech.edu/~xhou        http://bcmi.sjtu.edu.cn/~zhangliqing

# References

[1] V. Balasubramanian, D. Kimber, and M. Berry. Metabolically Efficient Information Processing. *Neural Computation*, 13(4):799–815, 2001.

[2] A. Bell and T. Sejnowski. The independent components of natural scenes are edge filters. *Vision Research*, 37(23):3327–3338, 1997.

[3] N. Bruce and J. Tsotsos. Saliency Based on Information Maximization. *Advances in Neural Information Processing Systems*, 18, 2006.

[4] D. Gao, V. Mahadevan, and N. Vasconcelos. The discriminant center-surround hypothesis for bottom-up saliency. pages 497–504, 2007.

[5] L. Itti and P. Baldi. Bayesian Surprise Attracts Human Attention. *Advances in Neural Information Processing Systems*, 18:547, 2006.

[6] L. Itti and C. Koch. Computational modeling of visual attention. *Nature Reviews Neuroscience*, 2(3):194–203, 2001.

[7] L. Itti, C. Koch, E. Niebur, et al. A model of saliency-based visual attention for rapid scene analysis. *IEEE Transactions on Pattern Analysis and Machine Intelligence*, 20(11):1254–1259, 1998.

[8] R. Klein. Inhibition of return. *Trends in Cognitive Sciences*, 4(4):138–147, 2000.

[9] C. Koch and T. Poggio. Predicting the visual world: silence is golden. *Nature Neuroscience*, 2:9–10, 1999.

[10] C. Koch and S. Ullman. Shifts in selective visual attention: towards the underlying neural circuitry. *Hum Neurobiol*, 4(4):219–27, 1985.

[11] W. Levy and R. Baxter. Energy Efficient Neural Codes. *Neural Codes and Distributed Representations: Foundations of Neural Computation*, 1999.

[12] S. Ling and M. Carrasco. When sustained attention impairs perception. *Nature neuroscience*, 9(10):1243, 2006.

[13] J. Maunsell and S. Treue. Feature-based attention in visual cortex. *Trends in Neurosciences*, 29(6):317–322, 2006.

[14] J. Najemnik and W. Geisler. Optimal eye movement strategies in visual search. *Nature*, 434(7031):387–391, 2005.

[15] B. Olshausen et al. Emergence of simple-cell receptive field properties by learning a sparse code for natural images. *Nature*, 381(6583):607–609, 1996.

[16] R. Peters and L. Itti. Beyond bottom-up: Incorporating task-dependent influences into a computational model of spatial attention. *IEEE Computer Society Conference on Computer Vision and Pattern Recognition*, 2007.

[17] R. Rao and D. Ballard. Predictive coding in the visual cortex: a functional interpretation of some extra-classical receptive-field effects. *Nature Neuroscience*, 2:79–87, 1999.

[18] J. Reynolds, T. Pasternak, and R. Desimone. Attention Increases Sensitivity of V4 Neurons. *Neuron*, 26(3):703–714, 2000.

[19] S. Treue and J. Maunsell. Attentional modulation of visual motion processing in cortical areas MT and MST. *Nature*, 382(6591):539–541, 1996.

[20] J. van Hateren. Real and optimal neural images in early vision. *Nature*, 360(6399):68–70, 1992.

[21] M. Wainwright. Visual adaptation as optimal information transmission. *Vision Research*, 39(23):3960–3974, 1999.

[22] D. Walther, D. Edgington, and C. Koch. Detection and tracking of objects in underwater video. *Computer Vision and Pattern Recognition, 2004. CVPR 2004. Proceedings of the 2004 IEEE Computer Society Conference on*, 1.

[23] D. Walther, U. Rutishauser, C. Koch, and P. Perona. Selective visual attention enables learning and recognition of multiple objects in cluttered scenes. *Computer Vision and Image Understanding*, 100(1-2):41–63, 2005.

[24] J. Wolfe, N. Klempen, and K. Dahlen. Post-attentive vision. *Journal of Experimental Psychology: Human Perception and Performance*, 26(2):693–716, 2000.
